# Optimal models of sound localization by barn owls

**Brian J. Fischer**
Division of Biology
California Institute of Technology
Pasadena, CA
`fischerb@caltech.edu`

## Abstract

Sound localization by barn owls is commonly modeled as a matching procedure where localization cues derived from auditory inputs are compared to stored templates. While the matching models can explain properties of neural responses, no model explains how the owl resolves spatial ambiguity in the localization cues to produce accurate localization for sources near the center of gaze. Here, I examine two models for the barn owl's sound localization behavior. First, I consider a maximum likelihood estimator in order to further evaluate the cue matching model. Second, I consider a maximum a posteriori estimator to test whether a Bayesian model with a prior that emphasizes directions near the center of gaze can reproduce the owl's localization behavior. I show that the maximum likelihood estimator can not reproduce the owl's behavior, while the maximum a posteriori estimator is able to match the behavior. This result suggests that the standard cue matching model will not be sufficient to explain sound localization behavior in the barn owl. The Bayesian model provides a new framework for analyzing sound localization in the barn owl and leads to predictions about the owl's localization behavior.

## 1 Introduction

Barn owls, the champions of sound localization, show systematic errors when localizing sounds. Owls localize broadband noise signals with great accuracy for source directions near the center of gaze [1]. However, localization errors increase as source directions move to the periphery, consistent with an underestimate of the source direction [1]. Behavioral experiments show that the barn owl uses the interaural time difference (ITD) for localization in the horizontal dimension and the interaural level difference (ILD) for localization in the vertical dimension [2]. Direct measurements of the sounds received at the ears for sources at different locations in space show that disparate directions are associated with very similar localization cues. Specifically, there is a similarity between ILD and ITD cues for directions near the center of gaze and directions with eccentric elevations on the vertical plane. How does the owl resolve this ambiguity in the localization cues to produce accurate localization for sound sources near the center of gaze?

Theories regarding the use of localization cues by the barn owl are drawn from the extensive knowledge of processing in the barn owl's auditory system. Neurophysiological and anatomical studies show that the barn owl's auditory system contains specialized circuitry that is devoted to extracting spectral ILD and ITD cues and processing them to derive source direction information [2]. It has been suggested that a spectral matching operation between ILD and ITD cues computed from auditory inputs and preferred ILD and ITD spectra associated with spatially selective auditory neurons underlies the derivation of spatial information from the auditory cues [3–6]. The spectral matching models reproduce aspects of neural responses, but none reproduces the sound localization behavior of the barn owl. In particular, the spectral matching models do not describe how the owl resolves ambiguities in the localization cues. In addition to spectral matching of localization cues, it is possible

that the owl incorporates prior experience or beliefs into the process of deriving direction estimates from the auditory input signals. These two approaches to sound localization can be formalized using the language of estimation theory as maximum likelihood (ML) and Bayesian solutions, respectively.

Here, I examine two models for the barn owl's sound localization behavior in order to further evaluate the spectral matching model and to test whether a Bayesian model with a prior that emphasizes directions near the center of gaze can reproduce the owl's localization behavior. I begin by viewing the sound localization problem as a statistical estimation problem. Maximum likelihood and maximum a posteriori (MAP) solutions to the estimation problem are compared with the localization behavior of a barn owl in a head turning task.

## 2 Observation model

To define the localization problem, we must specify an observation model that describes the information the owl uses to produce a direction estimate. Neurophysiological and behavioral experiments suggest that the barn owl derives direction estimates from ILD and ITD cues that are computed at an array of frequencies [2, 7, 8]. Note that when computed as a function of frequency, the ITD is given by an interaural phase difference (IPD).

Here I consider a model where the observation made by the owl is given by the ILD and IPD spectra derived from barn owl head-related transfer functions (HRTFs) after corruption with additive noise. For a source direction $(\theta, \phi)$, the observation vector $\mathbf{r}$ is expressed mathematically as

$$\mathbf{r} = \left[ \begin{array}{c} \mathbf{r}_{\mathrm{ILD}} \\ \mathbf{r}_{\mathrm{IPD}} \end{array} \right] = \left[ \begin{array}{c} \mathbf{ILD}_{\theta,\phi} \\ \mathbf{IPD}_{\theta,\phi} \end{array} \right] + \left[ \begin{array}{c} \boldsymbol{\eta}_{\mathrm{ILD}} \\ \boldsymbol{\eta}_{\mathrm{IPD}} \end{array} \right] \tag{1}$$

where the ILD spectrum $\mathbf{ILD}_{\theta,\phi} = [\mathrm{ILD}_{\theta,\phi}(\omega_1), \mathrm{ILD}_{\theta,\phi}(\omega_2), \ldots, \mathrm{ILD}_{\theta,\phi}(\omega_{N_f})]$ and the IPD spectrum $\mathbf{IPD}_{\theta,\phi} = [\mathrm{IPD}_{\theta,\phi}(\omega_1), \mathrm{IPD}_{\theta,\phi}(\omega_2), \ldots, \mathrm{IPD}_{\theta,\phi}(\omega_{N_f})]$ are specified at a finite number of frequencies. The ILD and IPD cues are computed directly from the HRTFs as

$$\mathrm{ILD}_{\theta,\phi}(\omega) = 20 \log_{10} \frac{|\hat{h}_{R(\theta,\phi)}(\omega)|}{|\hat{h}_{L(\theta,\phi)}(\omega)|} \tag{2}$$

and

$$\mathrm{IPD}_{\theta,\phi}(\omega) = \varphi_{R(\theta,\phi)}(\omega) - \varphi_{L(\theta,\phi)}(\omega), \tag{3}$$

where the left and right HRTFs are written as $\hat{h}_{L(\theta,\phi)}(\omega) = |\hat{h}_{L(\theta,\phi)}(\omega)|e^{i\varphi_{L(\theta,\phi)}(\omega)}$ and $\hat{h}_{R(\theta,\phi)}(\omega) = |\hat{h}_{R(\theta,\phi)}(\omega)|e^{i\varphi_{R(\theta,\phi)}(\omega)}$, respectively.

The noise corrupting the ILD spectrum is modeled as a Gaussian random vector with independent and identically distributed (i.i.d.) components, $\eta_{\mathrm{ILD}}(\omega_j) \sim \mathcal{N}(0, \sigma)$. The IPD spectrum noise vector is assumed to have i.i.d. components where each element has a von Mises distribution with parameter $\kappa$. The von Mises distribution can be viewed as a $2\pi$-periodic Gaussian distribution for large $\kappa$ and is a uniform distribution for $\kappa = 0$ [9]. I assume that the ILD and IPD noise terms are mutually independent.

With this noise model, the likelihood function has the form

$$p_{\mathbf{r}|\Theta,\Phi}(\mathbf{r}|\theta,\phi) = p_{\mathbf{r}_{\mathrm{ILD}}|\Theta,\Phi}(\mathbf{r}_{\mathrm{ILD}}|\theta,\phi)p_{\mathbf{r}_{\mathrm{IPD}}|\Theta,\Phi}(\mathbf{r}_{\mathrm{IPD}}|\theta,\phi) \tag{4}$$

where the ILD likelihood function is given by

$$p_{\mathbf{r}_{\mathrm{ILD}}|\Theta,\Phi}(\mathbf{r}_{\mathrm{ILD}}|\theta,\phi) = \frac{1}{(2\pi\sigma^2)^{N_f/2}} \exp[-\frac{1}{2\sigma^2} \sum_{j=1}^{N_f} (r_{\mathrm{ILD}}(\omega_j) - \mathrm{ILD}_{\theta,\phi}(\omega_j))^2] \tag{5}$$

and the IPD likelihood function is given by

$$p_{\mathbf{r}_{\mathrm{IPD}}|\Theta,\Phi}(\mathbf{r}_{\mathrm{IPD}}|\theta,\phi) = \frac{1}{(2\pi I_0(\kappa))^{N_f}} \exp[\kappa \sum_{j=1}^{N_f} \cos(r_{\mathrm{IPD}}(\omega_j) - \mathrm{IPD}_{\theta,\phi}(\omega_j))] \tag{6}$$

where $I_0(\kappa)$ is a modified Bessel function of the first kind of order $0$. The likelihood function will have peaks at directions where the expected spectral cues $\mathbf{ILD}_{\theta,\phi}$ and $\mathbf{IPD}_{\theta,\phi}$ are near the observed values $\mathbf{r}_{\mathrm{ILD}}$ and $\mathbf{r}_{\mathrm{IPD}}$.

# 3 Model performance measure

I evaluate maximum likelihood and maximum a posteriori methods for estimating the source direction from the observed ILD and IPD cues by computing an expected localization error and comparing the results to an owl's behavior. The performance of each estimation procedure at a given source direction is quantified by the expected absolute angular error $E[|\hat{\theta}(\mathbf{r}) - \theta| + |\hat{\phi}(\mathbf{r}) - \phi| \mid \theta, \phi]$. This measure of estimation error is directly compared to the behavioral performance of a barn owl in a head turning localization task [1]. The expected absolute angular error is approximated through Monte Carlo simulation as

$$E[|\hat{\theta}(\mathbf{r}) - \theta| + |\hat{\phi}(\mathbf{r}) - \phi| \mid \theta, \phi] \approx \mu(\{|\hat{\theta}(\mathbf{r}_i) - \theta|\}_{i=1}^N) + \mu(\{|\hat{\phi}(\mathbf{r}_i) - \phi|\}_{i=1}^N) \tag{7}$$

where the $\mathbf{r}_i$ are drawn from $p_{\mathbf{r}|\Theta,\Phi}(\mathbf{r}|\theta,\phi)$ and $\mu(\{\theta_i\}_{i=1}^N)$ is the circular mean of the angles $\{\theta_i\}_{i=1}^N$. The error is computed using HRTFs for two barn owls [10] and is calculated for directions in the frontal hemisphere with $5°$ increments in azimuth and elevation, as defined using double polar coordinates.

# 4 Maximum likelihood estimate

The maximum likelihood direction estimate is derived from the observed noisy ILD and IPD cues by finding the source direction that maximizes the likelihood function, yielding

$$(\hat{\theta}_{\text{ML}}(\mathbf{r}), \hat{\phi}_{\text{ML}}(\mathbf{r})) = \arg\max_{(\theta,\phi)} p_{\mathbf{r}|\Theta,\Phi}(\mathbf{r}|\theta,\phi). \tag{8}$$

This procedure amounts to a spectral cue matching operation. Each direction in space is associated with a particular ILD and IPD spectrum, as derived from the HRTFs. The direction with associated cues that are closest to the observed cues is designated as the estimate. This estimator is of particular interest because of the claim that salience in the neural map of auditory space in the barn owl can be described by a spectral cue matching operation [3, 4, 6].

The maximum likelihood estimator was unable to reproduce the owl's localization behavior. The performance of the maximum likelihood estimator depends on the two likelihood function parameters $\sigma$ and $\kappa$, which determine the ILD and IPD noise variances, respectively. For noise variances large enough that the error increased at peripheral directions, in accordance with the barn owl's behavior, the error also increased significantly for directions near the center of the interaural coordinate system (Figure 1). This pattern of error as a function of eccentricity, with a large central peak, is not consistent with the performance of the barn owl in the head turning task [1]. Additionally, directions near the center of gaze were often confused with directions in the periphery leading to a high variability in the direction estimates, which is not seen in the owl's behavior.

# 5 Maximum a posteriori estimate

In the Bayesian framework, the direction estimate depends on both the likelihood function and the prior distribution over source directions through the posterior distribution. Using Bayes' rule, the posterior density is proportional to the product of the likelihood function and the prior,

$$p_{\Theta,\Phi|\mathbf{r}}(\theta,\phi|\mathbf{r}) \propto p_{\mathbf{r}|\Theta,\Phi}(\mathbf{r}|\theta,\phi)p_{\Theta,\Phi}(\theta,\phi). \tag{9}$$

The prior distribution is used to summarize the owl's belief about the most likely source directions before an observation of ILD and IPD cues is made. Based on the barn owl's tendency to underestimate source directions [1], I use a prior that emphasizes directions near the center of gaze. The prior is given by a product of two one-dimensional von Mises distributions, yielding the probability density function

$$p_{\Theta,\Phi}(\theta,\phi) = \frac{\exp[\kappa_1 \cos(\theta) + \kappa_2 \cos(\phi)]}{(2\pi)^2 I_0(\kappa_1)I_0(\kappa_2)} \tag{10}$$

where $I_0(\kappa)$ is a modified Bessel function of the first kind of order $0$. The maximum a posteriori source direction estimate is computed for a given observation by finding the source direction that maximizes the posterior density, yielding

$$(\hat{\theta}_{\text{MAP}}(\mathbf{r}), \hat{\phi}_{\text{MAP}}(\mathbf{r})) = \arg\max_{(\theta,\phi)} p_{\Theta,\Phi|\mathbf{r}}(\theta,\phi|\mathbf{r}). \tag{11}$$

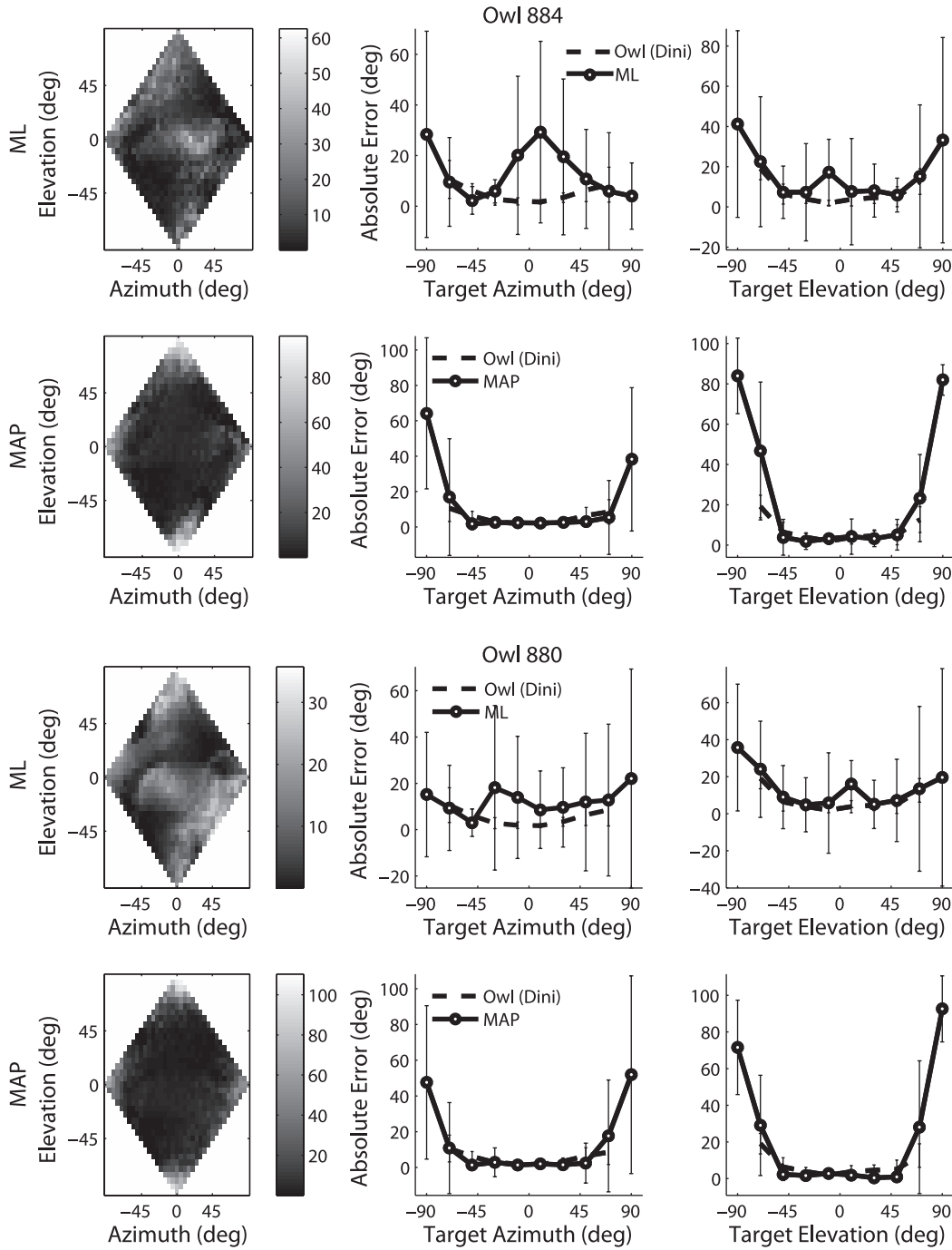

Figure 1: Estimation error in the model for the maximum likelihood (ML) and maximum a posteriori (MAP) estimates. HRTFs were used from owls 884 (top) and 880 (bottom). Left column: Estimation error at 685 locations in the frontal hemisphere plotted in double polar coordinates. Center column: Estimation error on the horizontal plane along with the estimation error of a barn owl in a head turning task [1]. Right column: Estimation error on the vertical plane along with the estimation error of a barn owl in a head turning task. Note that each plot uses a unique scale.

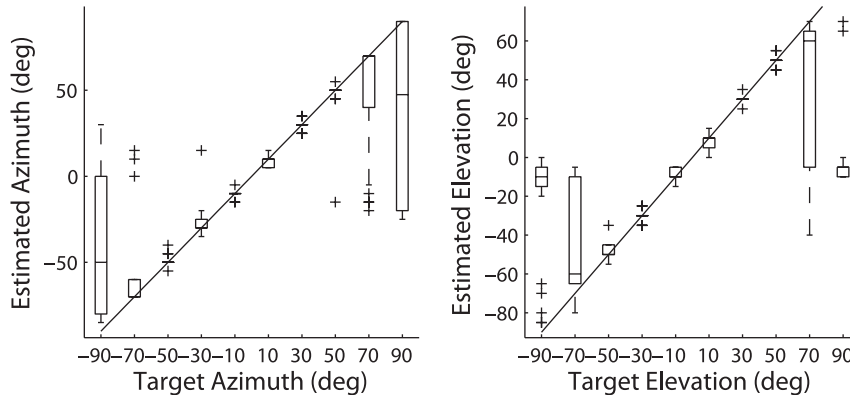

Figure 2: Estimates for the MAP estimator on the horizontal plane (left) and the vertical plane (right) using HRTFs from owl 880. The box extends from the lower quartile to the upper quartile of the sample. The solid line is the identity line. Like the owl, the MAP estimator underestimates the source direction.

In the MAP case, the estimate depends on spectral matching of observations with expected cues for each direction, but with a penalty on the selection of peripheral directions.

It was possible to find a MAP estimator that was consistent with the owl's localization behavior (Figures 1,2). For the example MAP estimators shown in Figures 1 and 2, the error was smallest in the central region of space and increased at the periphery. The largest errors occurred at the vertical extremes. This pattern of error qualitatively matches the pattern of error displayed by the owl in a head turning localization task [1].

The parameters that produced a behaviorally consistent MAP estimator correspond to a likelihood and prior with large variances. For the estimators shown in Figure 1, the likelihood function parameters were given by $\sigma = 11.5$ dB and $\kappa = 0.75$ for owl 880 and $\sigma = 10.75$ dB and $\kappa = 0.8$ for owl 884. For comparison, the range of ILD values normally experienced by the barn owl falls between $\pm 30$ dB [10]. The prior parameters correspond to an azimuthal width parameter $\kappa_1$ of 0.25 for owl 880 and 0.2 for owl 884 and an elevational width parameter $\kappa_2$ of 0.25 for owl 880 and 0.18 for owl 884.

The implication of this model for implementation in the owl's auditory system is that the spectral localization cues ILD and IPD do not need to be computed with great accuracy and the emphasis on central directions does not need to be large in order to produce the barn owl's behavior.

# 6 Discussion

## 6.1 A new approach to modeling sound localization in the barn owl

The simulation results show that the maximum likelihood model considered here can not reproduce the owl's behavior, while the maximum a posteriori solution is able to match the behavior. This result suggests that the standard spectral matching model will not be sufficient to explain sound localization behavior in the barn owl. Previously, suggestions have been made that sound localization by the barn owl can be described using the Bayesian framework [11, 12], but no specific models have been proposed. This paper demonstrates that a Bayesian model can qualitatively match the owl's localization behavior. The Bayesian approach described here provides a new framework for analyzing sound localization in the owl.

## 6.2 Failure of the maximum likelihood model

The maximum likelihood model fails because of the nature of spatial ambiguity in the ILD and IPD cues. The existence of spatial ambiguity has been noted in previous descriptions of barn owl HRTFs [3, 10, 13]. As expected, directions near each other have similar cues. In addition to sim-

ilarity of cues between proximal directions, distant directions can have similar ILD and IPD cues. Most significantly, there is a similarity between the ILD and IPD cues at the center of gaze and at peripheral directions on the vertical plane. The consequence of such ambiguity between distant directions is that noise in measuring localization cues can lead to large errors in direction estimation, as seen in the ML estimate. The results of the simulations suggest that a behaviorally accurate solution to the sound localization problem must include a mechanism that chooses between disparate directions which are associated with similar localization cues in such a way as to limit errors for source directions near the center of gaze. This work shows that a possible mechanism for choosing between such directions is to incorporate a bias towards directions at the center of gaze through a prior distribution and utilize the Bayesian estimation framework. The use of a prior that emphasizes directions near the center of gaze is similar to the use of central weighting functions in models of human lateralization [14].

## 6.3  Predictions of the Bayesian model

The MAP estimator predicts the underestimation of peripheral source directions on the horizontal and vertical planes (Figure 2). The pattern of error displayed by the MAP estimator qualitatively matches the owl's behavioral performance by showing increasing error as a function of eccentricity. Our evaluation of the model performance is limited, however, because there is little behavioral data for directions outside $\pm$ 70 deg [15,16]. For the owl whose performance is displayed in Figure 1, the largest errors on the vertical and horizontal planes were less than 20 deg and 11 deg, respectively. The model produces much larger errors for directions beyond 70 deg, especially on the vertical plane. The large errors in elevation result from the ambiguity in the localization cues on the vertical plane and the shape of the prior distribution. As discussed above, for broadband noise stimuli, there is a similarity between the ILD and IPD cues for central and peripheral directions on the vertical plane [3, 10, 13]. The presence of a prior distribution that emphasizes central directions causes direction estimates for both central and peripheral directions to be concentrated near zero deg. Therefore, estimation errors are minimal for sources at the center of gaze, but approach the magnitude of the source direction for peripheral source directions. Behavioral data shows that localization accuracy is the greatest near the center of gaze [1], but there is no data for localization performance at the most eccentric directions on the vertical plane. Further behavioral experiments must be performed to determine if the owl's error increases greatly at the most peripheral directions.

There is a significant spatial ambiguity in the localization cues when target sounds are narrowband. It is well known that spatial ambiguity arises from the way that interaural time differences are processed at each frequency [17–19]. The owl measures the interaural time difference for each frequency of the input sound as an interaural phase difference. Therefore, multiple directions in space that differ in their associated interaural time difference by the period of a tone at that frequency are consistent with the same interaural phase difference and can not be distinguished. Behavioral experiments show that the owl may localize a phantom source in the horizontal dimension when the signal is a tone [20]. Based on the presence of a prior that emphasizes directions near the center of gaze, I predict that for low frequency tones where phase equivalent directions lie near the center of gaze and at directions greater than 80 deg, confusion will always lead to an estimate of a source direction near zero degrees. This prediction can not be evaluated from available data because localization of tonal signals has only been systematically studied using 5 kHz tones with target directions at $\pm$ 20 deg [19]. Because the prior is broad, the target direction of $\pm$ 20 deg and the phantom direction of $\pm$ 50 deg may both be considered central.

The ILD cue also displays a significant ambiguity at high frequencies. At frequencies above 7 kHz, the ILD is non-monotonically related to the vertical position of a sound source [3, 10] (Figure 3). Therefore, for narrowband sounds, the owl can not uniquely determine the direction of a sound source from the ITD and ILD cues. I predict that for tonal signals above 7 kHz, there will be multiple directions on the vertical plane that are confused with directions near zero deg. I predict that confusion between source directions near zero deg and eccentric directions will always lead to estimates of directions near zero deg. There is no available data to evaluate this prediction.

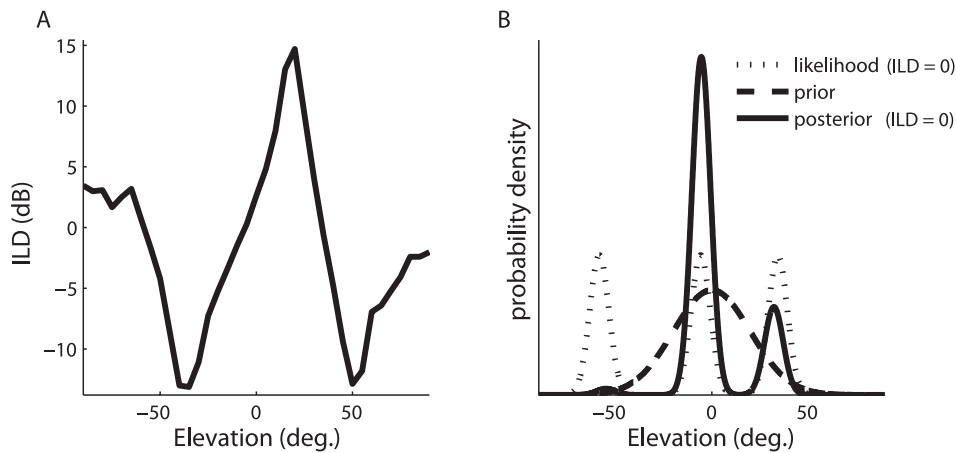

Figure 3: Model predictions for localization of tones on the vertical plane. (A) ILD as a function of elevation at 8 kHz, computed from HRTFs of owl 880 recorded by Keller et al. (1998). (B) Given an ILD of 0 dB, a likelihood function (dots) based on matching cues to expected values would be multimodal with three equal peaks. If the target is at any of the three directions, there will be large localization errors because of confusion with the other directions. If a prior emphasizing frontal space (dashed) is included, a posterior density equal to the product of the likelihood and the prior would have a main peak at 0 deg elevation. Using a maximum a posteriori estimate, large errors would be made if the target is above or below. However, few errors would be observed when the target is near 0 deg.

## 6.4   Testing the Bayesian model

Further head turning localization experiments with barn owls must be performed to test predictions generated by the Bayesian hypothesis and to provide constraints on a model of sound localization. Experiments should test the localization accuracy of the owl for broadband noise sources and tonal signals at directions covering the frontal hemisphere. The Bayesian model will be supported if, first, localization accuracy is high for both tonal and broadband noise sources near the center of gaze and, second, peripherally located sources are confused for targets near the center of gaze, leading to large localization errors. Additionally, a Bayesian model should be fit to the data, including points away from the horizontal and vertical planes, using a nonparametric prior [21, 22]. While the model presented here, using a von Mises prior, qualitatively matches the performance of the owl, the performance of the Bayesian model may be improved by removing assumptions about the structure of the prior distribution.

## 6.5   Implications for neural processing

The analysis presented here does not directly address the neural implementation of the solution to the localization problem. However, our abstract analysis of the sound localization problem has implications for neural processing. Several models exist that reproduce the basic properties of ILD, ITD, and space selectivity in ICx and OT neurons using a spectral matching procedure [3, 5, 6]. These results suggest that a Bayesian model is not necessary to describe the responses of individual ICx and OT neurons. It may be necessary to look in the brainstem motor targets of the optic tectum to find neurons that resolve the ambiguity present in sound stimuli and show responses that reflect the MAP solution. This implies that the prior distribution is not employed until the final stage of processing. The prior may correspond to the distribution of best directions of space-specific neurons in ICx and OT, which emphasizes directions near the center of gaze [23].

## 6.6   Conclusion

This analysis supports the Bayesian model of the barn owl's solution to the localization problem over the maximum likelihood model. This result suggests that the standard spectral matching model will not be sufficient to explain sound localization behavior in the barn owl. The Bayesian model

provides a new framework for analyzing sound localization in the owl. The simulation results using the MAP estimator lead to testable predictions that can be used to evaluate the Bayesian model of sound localization in the barn owl.

### Acknowledgments

I thank Kip Keller, Klaus Hartung, and Terry Takahashi for providing the head-related transfer functions and Mark Konishi and José Luis Peña for comments and support.

### References

[1] E.I. Knudsen, G.G. Blasdel, and M. Konishi. Sound localization by the barn owl (*Tyto alba*) measured with the search coil technique. *J. Comp. Physiol.*, 133:1–11, 1979.

[2] M. Konishi. Coding of auditory space. *Annu. Rev. Neurosci.*, 26:31–55, 2003.

[3] M.S. Brainard, E.I. Knudsen, and S.D. Esterly. Neural derivation of sound source location: Resolution of spatial ambiguities in binaural cues. *J. Acoust. Soc. Am.*, 91(2):1015–1027, 1992.

[4] B.J. Arthur. Neural computations leading to space-specific auditory responses in the barn owl. Ph.D. thesis, Caltech, 2001.

[5] B.J. Fischer. A model of the computations leading to a representation of auditory space in the midbrain of the barn owl. D.Sc. thesis, Washington University in St. Louis, 2005.

[6] C.H. Keller and T.T. Takahashi. Localization and identification of concurrent sounds in the owl's auditory space map. *J. Neurosci.*, 25:10446–10461, 2005.

[7] I. Poganiatz and H. Wagner. Sound-localization experiments with barn owls in virtual space: influence of broadband interaural level difference on head-turning behavior. *J. Comp. Physiol. A*, 187:225–233, 2001.

[8] D.R. Euston and T.T. Takahashi. From spectrum to space: The contribution of level difference cues to spatial receptive fields in the barn owl inferior colliculus. *J. Neurosci.*, 22(1):284–293, Jan. 2002.

[9] Evans M., Hastings N., and Peacock B. von Mises Distribution. In *Statistical Distributions, 3rd ed.*, pages 189–191. Wiley, New York, 2000.

[10] C.H. Keller, K. Hartung, and T.T. Takahashi. Head-related transfer functions of the barn owl: measurement and neural responses. *Hearing Research*, 118:13–34, 1998.

[11] R.O. Duda. *Elevation dependence of the interaural transfer function*, chapter 3 in Binaural and Spatial Hearing in Real and Virtual Environments, pages 49–75. New Jersey: Lawrence Erlbaum Associates, 1997.

[12] Witten I.B. and Knudsen E.I. Why seeing is believing: Merging auditory and visual worlds. *Neuron*, 48:489–496, 2005.

[13] J.F Olsen, E.I. Knudsen, and S.D. Esterly. Neural maps of interaural time and intensity differences in the optic tectum of the barn owl. *J. Neurosci.*, 9:2591–2605, 1989.

[14] R.M. Stern and H.S. Colburn. Theory of binaural interaction based on auditory-nerve data. IV. A model for subjective lateral position. *J. Acoust. Soc. Am.*, 64:127–140, 1978.

[15] H. Wagner. Sound-localization deficits induced by lesions in the barn owl's auditory space map. *J. Neurosci.*, 13:371–386, 1993.

[16] I. Poganiatz, I. Nelken, and H. Wagner. Sound-localization experiments with barn owls in virtual space: influence of interaural time difference on head-turning behavior. *J. Ass. Res. Otolarnyg.*, 2:1–21, 2001.

[17] T. Takahashi and M. Konishi. Selectivity for interaural time difference in the owl's midbrain. *J. Neurosci.*, 6(12):3413–3422, 1986.

[18] J.A. Mazer. How the owl resolves auditory coding ambiguity. *Proc. Natl. Acad. Sci. USA*, 95:10932–10937, 1998.

[19] K. Saberi, Y. Takahashi, H. Farahbod, and M. Konishi. Neural bases of an auditory illusion and its elimination in owls. *Nature Neurosci.*, 2(7):656–659, 1999.

[20] E.I. Knudsen and M. Konishi. Mechanisms of sound localization in the barn owl (*Tyto alba*) measured with the search coil technique. *J. Comp. Phys. A*, (133):13–21, 1979.

[21] Liam Paninski. Nonparametric inference of prior probabilities from Bayes-optimal behavior. In Y. Weiss, B. Schölkopf, and J. Platt, editors, *Advances in Neural Information Processing Systems 18*, pages 1067–1074. MIT Press, Cambridge, MA, 2006.

[22] Stocker A.A. and Simoncelli E.P. Noise characteristics and prior expectations in human visual speed perception. *Nature Neurosci.*, 9(4):578–585, 2006.

[23] E.I. Knudsen and M. Konishi. A neural map of auditory space in the owl. *Science*, 200:795–797, 1978.

